# Measures of Clustering Quality: A Working Set of Axioms for Clustering

**Margareta Ackerman and Shai Ben-David**
School of Computer Science
University of Waterloo, Canada

## Abstract

Aiming towards the development of a general clustering theory, we discuss abstract axiomatization for clustering. In this respect, we follow up on the work of Kleinberg, ([1]) that showed an impossibility result for such axiomatization. We argue that an impossibility result is not an inherent feature of clustering, but rather, to a large extent, it is an artifact of the specific formalism used in [1].

As opposed to previous work focusing on clustering functions, we propose to address clustering quality measures as the object to be axiomatized. We show that principles like those formulated in Kleinberg's axioms can be readily expressed in the latter framework without leading to inconsistency.

A clustering-quality measure (CQM) is a function that, given a data set and its partition into clusters, returns a non-negative real number representing how strong or conclusive the clustering is. We analyze what clustering-quality measures should look like and introduce a set of requirements (axioms) for such measures. Our axioms capture the principles expressed by Kleinberg's axioms while retaining consistency.

We propose several natural clustering quality measures, all satisfying the proposed axioms. In addition, we analyze the computational complexity of evaluating the quality of a given clustering and show that, for the proposed CQMs, it can be computed in polynomial time.

## 1 Introduction

In his highly influential paper, [1], Kleinberg advocates the development of a theory of clustering that will be "independent of any particular algorithm, objective function, or generative data model." As a step in that direction, Kleinberg sets up a set of "axioms" aimed to define what a clustering function is. Kleinberg suggests three axioms, each sounding plausible, and shows that these seemingly natural axioms lead to a contradiction - there exists no function that satisfies all three requirements.

Kleinberg's result is often interpreted as stating the impossibility of defining what clustering is, or even of developing a general theory of clustering. We disagree with this view. In this paper we show that the impossibility result is, to a large extent, due to the specific formalism used by Kleinberg rather than being an inherent feature of clustering.

Rather than attempting to define what a *clustering function* is, and demonstrating a failed attempt, as [1] does, we turn our attention to the closely related issue of evaluating the *quality of a given data clustering*. In this paper we develop a formalism and a consistent axiomatization of that latter notion.

As it turns out, the *clustering-quality* framework is richer and more flexible than that of clustering *functions*. In particular, it allows the postulation of axioms that capture the features that Kleinberg's axioms aim to express, without leading to a contradiction.

A *clustering-quality measure* is a function that maps pairs of the form $(dataset, clustering)$ to some ordered set (say, the set of non-negative real numbers), so that these values reflect how 'good' or 'cogent' that clustering is.

Measures for the quality of a clusterings are of interest not only as a vehicle for axiomatizing clustering. The need to measure the quality of a given data clustering arises naturally in many clustering issues. The aim of clustering is to uncover meaningful groups in data. However, not any arbitrary partitioning of a given data set reflects such a structure. Upon obtaining a clustering, usually via some algorithm, a user needs to determine whether this clustering is sufficiently meaningful to rely upon for further data mining analysis or practical applications. Clustering-quality measures (CQMs) aim to answer that need by quantifying how good is any specific clustering.

Clustering-quality measures may also be used to help in clustering model-selection by comparing different clusterings over the same data set (e.g., comparing the results of a given clustering paradigm over different choices of clustering parameters, such as the number of clusters).

When posed with the problem of finding a clustering-quality measure, a first attempt may be to invoke the loss (or objective) function used by a clustering algorithm, such as $k$-means or $k$-median, as a clustering-quality measure. However, such measures have some shortcomings for the purpose at hand. Namely, these measures are usually not scale-invariant, and they cannot be used to compare the quality of clusterings obtained by different algorithms aiming to minimize different clustering costs (e.g., $k$-means with different values of $k$). See Section 6 for more details.

Clustering quality has been previously discussed in the applied statistics literature, where a variety of techniques for evaluating *'cluster validity'* were proposed. Many of these methods, such as the external criteria discussed in [2], are based on assuming some predetermined data generative model, and as such do not answer our quest for a general theory of clustering. In this work, we are concerned with quality measures regardless of any specific generative model, for examples, see the internal criteria surveyed in [2].

We formulate a theoretical basis for clustering-quality evaluations. We propose a set of requirements ('axioms') of clustering-quality measures. We demonstrate the relevance and consistency of these axioms by showing that several natural notions satisfy these requirements. In particular, we introduce quality-measures that reflect the underlying intuition of center-based and linkage-based clustering. These notions all satisfy our axioms, and, given a data clustering, their value on that clustering can be computed in polynomial time.

*Paper outline*: we begin by presenting Kleinberg's axioms for clustering functions and discuss their failure. In Section 4.3 we show how these axioms can be translated into axioms pertaining clustering quality measures, and prove that the resulting set of axioms is consistent. In Section 4, we discuss desired properties of an axiomatization and propose an accordingly revised set of axioms. Next, in Section 5 we present several clustering-quality measures, and claim that they all satisfy our axioms. Finally, in Section 5.3, we show that the quality of a clustering can be computed in polynomial time with respect to our proposed clustering-quality measures.

## 2 Definitions and Notation

Let $X$ be some domain set (usually finite). A function $d : X \times X \rightarrow \mathbf{R}$ is a *distance function* over $X$ if $d(x_i, x_i) \geq 0$ for all $x_i \in X$, for any $x_i, x_j \in X$, $d(x_i, x_j) > 0$ if and only if $x_i \neq x_j$, and $d(x_i, x_j) = d(x_j, x_i)$ otherwise. Note that we do not require the triangle inequality.

A *$k$-clustering* of $X$ is a $k$-partition, $C = \{C_1, C_2, \ldots, C_k\}$. That is, $C_i \cap C_j = \emptyset$ for $i \neq j$ and $\cup_{i=1}^{k} C_i = X$. A *clustering* of $X$ is a $k$-clustering of $X$ for some $k \geq 1$. A clustering is *trivial* if each of its clusters contains just one point, or if it consists of just one cluster.

For $x, y \in X$ and clustering $C$ of $X$, we write $x \sim_C y$ whenever $x$ and $y$ are in the same cluster of clustering $C$ and $x \not\sim_C y$, otherwise.

A *clustering function* for some domain set $X$ is a function that takes a distance function $d$ over $X$, and outputs a clustering of $X$.

A *clustering-quality measure* (CQM) is a function that is given a clustering $C$ over $(X, d)$ (where $d$ is a distance function over $X$) and returns a non-negative real number, as well as satisfies some additional requirements. In this work we explore the question of what these requirements should be.

# 3 Kleinberg's Axioms

Kleinberg, [1], proposes the following three axioms for clustering functions. These axioms are intended to capture the meaning of clustering by determining which functions (from a domain set endowed with a distance function) are worthy of being considered *clustering functions* and which are not. Kleinberg shows that the set is inconsistent - there exist no functions that satisfies all three axioms.

The first two axioms require invariance of the clustering that $f$ defines under some changes of the input distance function.

**Function Scale Invariance**: Scale invariance requires that the output of a clustering function be invariant to uniform scaling of the input.

*A function $f$ is* scale-invariant *if for every distance function $d$ and positive $\lambda$, $f(d) = f(\lambda d)$ (where $\lambda d$ is defined by setting, for every pair of domain points $x, y$, $\lambda d(x, y) = \lambda \cdot d(x, y)$).*

**Function Consistency**: Consistency requires that if within-cluster distances are decreased, and between-cluster distances are increased, then the output of a clustering function does not change. Formally,

- Given a clustering $C$ over $(X, d)$, a distance function $d'$ is a *C-consistent variant* of $d$, if $d'(x, y) \leq d(x, y)$ for all $x \sim_C y$, and $d'(x, y) \geq d(x, y)$ for all $x \not\sim_C y$.
- *A function $f$ is* consistent *if $f(d) = f(d')$ whenever $d'$ is an $f(d)$-consistent variant of $d$.*

**Function Richness**: Richness requires that by modifying the distance function, any partition of the underlying data set can be obtained.

*A function $f$ is* rich *if for each partitioning, $C$, of $X$, there exists a distance function $d$ over $X$ so that $f(d) = C$.*

**Theorem 1 (Kleinberg, [1])** *There exists no clustering function that simultaneously satisfies scale invariance, consistency and richness.*

*Discussion:* The intuition behind these axioms is rather clear. Let us consider, for example, the Consistency requirement. It seems reasonable that by pulling closer points that are in the same cluster and pushing further apart points in different clusters, our confidence in the given clustering will only rise. However, while this intuition can be readily formulated in terms of clustering quality (namely, "changes as these should not decrease the quality of a clustering"), the formulation through clustering functions says more. It actually requires that such changes to the underlying distance function should not create any new contenders for the best-clustering of the data.

For example, consider Figure 1, where we illustrate a good 6-clustering. On the right hand-side, we show a consistent change of this 6-clustering. Notice that the resulting data has a 3-clustering that is reasonably better than the original 6-clustering. While one may argue that the quality of the original 6-clustering has not decreased as a result of the distance changes, the quality of the 3-clustering has improved beyond that of the 6-clustering. This illustrates a significant weakness of the consistency axiom for clustering *functions*.

The implicit requirement that the original clustering remains the best clustering following a consistent change is at the heart of Kleinberg's impossibility result. As we shall see below, once we relax that extra requirement the axioms are no longer unsatisfiable.

# 4 Axioms of Clustering-Quality Measures

In this section we change the primitive that is being defined by the axioms from clustering functions to clustering-quality measures (CQM). We reformulate the above three axioms in terms of CQMs

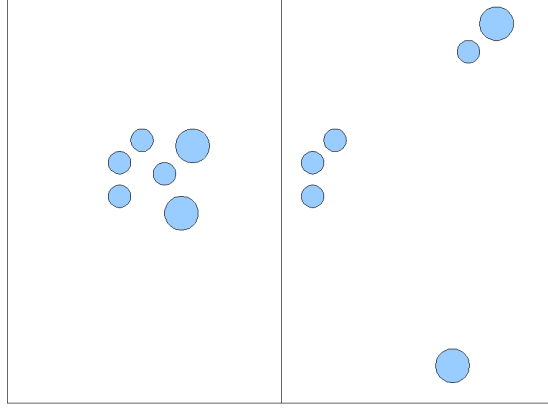

Figure 1: A consistent change of a $6$-clustering.

and show that this revised formulation is not only consistent, but is also satisfied by a number of natural clustering quality measures. In addition, we extend the set of axioms by adding another axiom (of clustering-quality measures) that is required to rule out some measures that should not be counted as CQMs.

## 4.1 Clustering-Quality Measure Analogues to Kleinberg's Axioms

The translation of the Scale Invariance axiom to the CQM terminology is straightforward:

**Definition 1 (Scale Invariance)** *A quality measure $m$ satisfies* scale invariance *if for every clustering $C$ of $(X, d)$, and every positive $\lambda$, $m(C, X, d) = m(C, X, \lambda d)$.*

The translation of the Consistency axiom is the place where the resulting CQM formulation is indeed weaker than the original axiom for functions. While it clearly captures the intuition that consistent changes to $d$ should not hurt the quality of a given partition, it allows the possibility that, as a result of such a change, some partitions will improve more than others[1].

**Definition 2 (Consistency)** *A quality measure $m$ satisfies* consistency *if for every clustering $C$ over $(X, d)$, whenever $d'$ is a $C$ consistent variant of $d$, then $m(C, X, d') \geq m(C, X, d)$.*

**Definition 3 (Richness)** *A quality measure $m$ satisfies* richness *if for each non-trivial clustering $C$ of $X$, there exists a distance function $d$ over $X$ such that $C = Argmax\{m(C, X, d)\}$.*

**Theorem 2** *Consistency, scale invariance, and richness for clustering-quality measures form a consistent set of requirements.*

*Proof:* To show that scale-invariance, consistency, and richness form a consistent set of axioms, we present a clustering quality measure that satisfies the three axioms. This measure captures a quality that is intuitive for center-based clusterings. In Section 5, we introduce more quality measures that capture the goal of other types of clusterings. All of these CQM's satisfy the above three axioms.

For each point in the data set, consider the ratio of the distance from the point to its closest center to the distance from the point to its second closest center. Intuitively, the smaller this ratio is, the better the clustering (points are 'more confident' about their cluster membership). We use the average of this ratio as a quality measure.

**Definition 4 (Relative Point Margin)** *The $\mathcal{K}$-Relative Point Margin of $x \in X$ is $\mathcal{K}\text{-}RM_{X,d}(x) = \frac{d(x,c_x)}{d(x,c_{x'})}$, where $c_x \in \mathcal{K}$ is the closest center to $x$, $c_{x'} \in \mathcal{K}$ is a second closest center to $x$, and $\mathcal{K} \subseteq X$.*

A set $\mathcal{K}$ is a *representative set* of a clustering $C$ if it consists of exactly one point from each cluster of $C$.

**Definition 5 (Representative Set)** *A set $\mathcal{K}$ is a representative set of clustering $C = \{C_1, C_2, \ldots, C_k\}$ if $|\mathcal{K}| = k$ and for all $i$, $\mathcal{K} \cap C_i \neq \emptyset$.*

**Definition 6 (Relative Margin)** *The Relative Margin of a clustering $C$ over $(X, d)$ is*

$$RM_{X,d}(C) = \min_{\mathcal{K} \text{ is a representative set of } C} avg_{x \in X \setminus \mathcal{K}} \mathcal{K}\text{-}RM_{X,d}(x).$$

Smaller values of Relative Margin indicate better clustering quality.

**Lemma 1** *Relative Margin is scale-invariant.*

*proof:* Let $C$ be a clustering of $(X, d)$. Let $d'$ be a distance function so that $d'(x, y) = \alpha d(x, y)$ for all $x, y \in X$ and some $\alpha \in \mathbf{R}^+$. Then for any points $x, y, z \in X$, $\frac{d(x,y)}{d(x,z)} = \frac{d'(x,y)}{d'(x,z)}$. Note also that scaling does not change the centers selected by Relative Margin. Therefore, $RM_{X,d'}(C) = RM_{X,d}(C)$.

**Lemma 2** *Relative Margin is consistent.*

*proof:* Let $C$ be a clustering of distance function $(X, d)$. Let $d'$ be a $C$ consistent variant of $d$. Then for $x \sim_C y$, $d'(x, y) \leq d(x, y)$ and for $x \not\sim_C y$, $d'(x, y) \geq d(x, y)$. Therefore, $RM_{X,d'}(C) \leq RM_{X,d}(C)$.

**Lemma 3** *Relative Margin is rich.*

*proof:* Given a non-trivial clustering $C$ over a data set $X$, consider the distance function $d$ where $d(x, y) = 1$ for all $x \sim_C y$, and $d(x, y) = 10$ for all $x \not\sim_C y$. Then $C = Argmin\{m(C, X, d)\}$.

It follows that scale-invariance, consistency, and richness are consistent axioms.

### 4.2 Soundness and Completeness of Axioms

What should a set of "axioms for clustering" satisfy? Usually, when a set of axioms is proposed for some semantic notion (or a class of objects, say clustering functions), the aim is to have both *soundness* and *completeness*. Soundness means that every element of the described class satisfies all axioms (so, in particular, soundness implies consistency of the axioms), and completeness means that every property shared by all objects of the class is implied by the axioms. Intuitively, ignoring logic subtleties, a set of axioms is complete for a class of objects if any element outside that class fails at least one of these axioms.

In our context, there is a major difficulty - there exist no semantic definition of what clustering is. We wish to use the axioms as a *definition* of clustering functions, but then what is the meaning of soundness and completeness? We have to settle for less. While we do not have a clear definition of what is clustering and what is not, we do have some examples of functions that should be considered clustering functions, and we can come up with some examples of partitionings that are clearly not worthy of being called "clustering". We replace soundness by the requirement that all of our axioms are satisfied by all these examples of common clustering functions (relaxed soundness), and we want that partitioning functions that are clearly not clusterings fail at least one of our axioms (relaxed completeness).

In this respect, the axioms of [1] badly fail (the relaxed version of) soundness. For each of these axioms there are natural clustering functions that fail to satisfy it (this is implied by Kleinberg's demonstration that any pair of axioms is satisfied by a natural clustering, while the three together never hold). We argue that our scale invariance, consistency, and richness, are sound for the class of CQMs. However, they do not make a complete set of axioms, even in our relaxed sense. There are functions that should not be considered "reasonable clustering quality measures" and yet they satisfy these three axioms. One type of "non-clustering-functions" are functions that make cluster membership decisions based on the identity of domain points. For example, the function that returns

the Relative Margin of a data set whenever some specific pair of data points belong to the same cluster, and twice the Relative Margin of the data set otherwise. We overcome this problem by introducing a new axiom.

### 4.3 Isomorphism Invariance

This axiom resembles the *permutation invariance* objective function axiom by Puzicha et al. [3], modeling the requirement that clustering should be indifferent to the individual identity of clustered elements. This axiom of clustering-quality measures does not have a corresponding Kleinberg axiom.

**Definition 7 (Clustering Isomorphism)** *Two clusterings $C$ and $C'$ over the same domain, $(X, d)$, are* isomorphic*, denoted $C \approx_d C'$, if there exists a distance-preserving isomorphism $\phi : X \to X$, such that for all $x, y \in X$, $x \sim_C y$ if and only if $\phi(x) \sim_{C'} \phi(y)$.*

**Definition 8 (Isomorphism Invariance)** *A quality measure $m$ is* isomorphism -invariant *if for all clusterings $C$, $C'$ over $(X, d)$ where $C \approx_d C'$, $m(C, X, d) = m(C', X, d)$.*

**Theorem 3** *The set of axioms consisting of Isomorphism Invariance, Scale Invariance, Consistency, and Richness, (all in their CQM formulation) is a consistent set of axioms.*

*Proof:* Just note that the Relative Margin quality measure satisfies all four axioms.

As mentioned in the above discussion, to have a satisfactory axiom system, for any notion, one needs to require more than just consistency. To be worthy of being labeled 'axioms', the requirements we propose should be satisfied by *any* reasonable notion of CQM. Of course, since we cannot define what CQMs are "reasonable", we cannot turn this into a formal statement. What we can do, however, is demonstrate that a variety of natural CQMs do satisfy all our axioms. This is done in the next section.

## 5 Examples of Clustering Quality Measures

In a survey of validity measures, Milligan [2] discusses examples of quality measures that satisfy our axioms (namely, scale-invariance, consistency, richness, and perturbation invariance). We have verified that the best performing internal criteria examined in [2], satisfy all our axioms.

In this section, we introduce two novel QCMs; a measure that reflects the underlying intuition of linkage-based clustering, and a measure for center-based clustering. In addition to satisfying the axioms, given a clustering, these measures can computed in polynomial time.

### 5.1 Weakest Link

In linkage-based clustering, whenever a pair of points share the same cluster they are connected via a tight chain of points in that cluster. The weakest link quality measure focuses on the longest link in such a chain.

**Definition 9 (Weakest Link Between Points)**
$$C\text{-}WL_{X,d}(x,y) = \min_{x_1, x_2, \ldots, x_\ell \in C_i} (\max(d(x, x_1), d(x_1, x_2), \ldots, d(x_\ell, y))),$$
*where $C$ is a clustering over $(X, d)$ and $C_i$ is a cluster in $C$.*

The weakest link of $C$ is the maximal value of $C\text{-}WL_{X,d}(x,y)$ over all pairs of points belonging to the same cluster, divided by the shortest between-cluster distance.

**Definition 10 (Weakest Link of $C$)** *The* Weakest Link *of a clustering $C$ over $(X, d)$ is*
$$WL(C) = \frac{\max_{x \sim_C y} C\text{-}WL_{X,d}(x,y)}{\min_{x \not\sim_C y} d(x,y)}.$$

The range of values of weakest link is $(0, \infty)$.

## 5.2 Additive Margin

In Section 4.3, we introduced Relative Margin, a quality measure for center-based clustering. We now introduce another quality measure for center-based clustering. Instead of looking at ratios, Additive Margin evaluates differences.

**Definition 11 (Additive Point Margin)** *The $\mathcal{K}$-Additive Point Margin of $x$ is $\mathcal{K}\text{-}AM_{X,d}(x) = d(x, c_{x'}) - d(x, c_x)$, where $c_x \in \mathcal{K}$ is the closest center to $x$, $c_{x'} \in \mathcal{K}$ is a second closest center to $x$, and $\mathcal{K} \subseteq X$.*

The Additive Margin of a clustering is the average Additive Point Margin, divided by the average within-cluster distance. The normalization is necessary for scale invariance.

**Definition 12 (Additive Margin)** *The Additive Margin of a center-based clustering $C$ over $(X, d)$ is*

$$AM_{X,d}(C) = \min_{\mathcal{K} \text{ is a representative set of } C} \frac{\frac{1}{|X|} \sum_{x \in X} \mathcal{K}\text{-}AM_{X,d}(x)}{\frac{1}{|\{\{x,y\} \subseteq X | x \sim_C y\}|} \sum_{x \sim_C y} d(x, y)}.$$

Unlike Relative Margin, Additive Margin gives higher values to better clusterings.

## 5.3 Computational complexity

For a clustering-quality measure to be useful, it is important to be able to quickly compute the quality of a clustering using that measure. The quality of a clustering using the measures presented in this paper can be computed in polynomial time in terms of $n$ (the number of points in the data set).

Using relative or Additive Margin, it takes $O(n^{k+1})$ operations to compute the clustering quality of a data set, which is exponential in $k$. If a set of centers is given, the Relative Margin can be computed in $O(nk)$ operations and the Additive Margin can be computed in $O(n^2)$ operations. The weakest link of a clustering can be computed in $O(n^3)$ operations.

## 5.4 Variants of quality measures

Given a clustering-quality measure, we can construct new quality measures with different characteristics by applying the quality measure on a subset of clusters. It suffices to consider a quality measure $m$ that is defined for clusterings consisting of 2 clusters. Given such measure, we can create new quality measures. For example,

$$m_{min}(C, X, d) = \min_{S \subseteq C, |S|=2} m(S, X, d),$$

measures the worst quality of a pair of clusters in $C$.

Alternately, we can define, $m_{max}(C, X, d)$ and $m_{avg}(C, X, d)$, which evaluate the best or average quality of a pair of clusters in $C$. A nice feature of these variations is that if $m$ satisfies the four axioms of clustering-quality measures then so do $m_{min}$, $m_{max}$, and $m_{avg}$.

More generally, if $m$ is defined for clusterings on an arbitrary number of clusters, we can define a quality measure as a function of $m$ over larger clusterings. For example, $m_{max\ subset}(C, X, d) = \max_{S \subseteq C, |S| \geq 2} m(S, X, d)$. Many such variations, which apply existing clustering-quality measures on subsets of clusters, satisfy the axioms of clustering-quality measures whenever the original quality measure satisfies the axioms.

# 6 Dependence on Number of Clusters

The clustering-quality measures discussed in this paper up to now are independent of the number of clusters, which enables the comparison of clusterings with a different number of clusters. In this section we discuss an alternative type of clustering quality evaluation, that depends on the number of clusters. Such quality measures arise naturally from common loss functions (or, objective functions) that drive clustering algorithm, such as $k$-means or $k$-median.

These common loss functions fail to satisfy two of our axioms, scale-invariance and richness. One can easily overcome the dependence on scaling by normalization. As we will show, the resulting normalized loss functions make a different type of clustering-quality measures from the measures we previously discussed, due to their dependence on the number of clusters.

A natural remedy to the failure of scale invariance is to normalize a loss function by dividing it by the variance of the data, or alternatively, by the loss of the 1-clustering of the data.

**Definition 13 ($\mathcal{L}$-normalization)** *The $\mathcal{L}$-normalization of a clustering $C$ over $(X, d)$ is*

$$\mathcal{L}\text{-}normalize(C, X, d) = \frac{\mathcal{L}(C_{all}, X, d)}{\mathcal{L}(C, X, d)}.$$

*where $C_{all}$ denotes the $1$-clustering of $X$.*

Common loss functions, even after normalization, usually have a bias towards either more refined or towards more coarse clusterings – they assign lower cost (that is, higher quality) to more refined (respectively, coarse) clusterings. This prevents using them as a meaningful tool for comparing the quality of clusterings with different number of clusters. We formalize this feature of common clustering loss functions through the notion of *refinement preference*:

**Definition 14 (Refinement and coarsening)** *For a pair of clusterings $C, C'$ of the same domain, we say $C'$ is a* refinement *of $C$ (or, equivalently, that $C$ is a coarsening of $C'$) if for every cluster $C_i$ of $C$, $C_i$ is a union of clusters of $C'$.*

**Definition 15 (Refinement/Coarsening Preference)** *A measure $m$ is* refinement-preferring *if for every clustering $C$ of $(X, d)$ if it has a non-trivial refinement, then there exists such a refinement $C'$ of $C$ for which $m(C', X, d) > m(C, X, d)$.* Coarsening-preferring *measures are defined analogously.*

Note that both refinement preferring and coarsening preferring measures fail to satisfy the Richness axiom.

It seems that there is a divide between two types of evaluations for clusterings; those that satisfy richness, and those that satisfy either refinement or coarsening preference. To evaluate the quality of a clustering using a refinement (and coarsening) preferring measure, it is essential to fix the number of clusters. Since the correct number of clusters is often unknown, measures that are independent of the number of clusters apply in a more general setting.

## 7   Conclusions

We have investigated the possibility of providing a general axiomatic basis for clustering. Our starting point was the impossibility result of Kleinberg. We argue that a natural way to overcome these negative conclusions is by changing the primitive used to formulate the axioms from clustering functions to clustering quality measures (CQMs). We demonstrate the merits of the latter framework by providing a set of axioms for CQMs that captures the essence of all of Kleinberg's axioms while maintaining consistency. We propose several CQMs that satisfy our proposed set of axioms. We hope that this work, and our demonstration of a way to overcome the "impossibility result" will stimulate further research towards a general theory of clustering.

## Footnotes

[1]The following formalization assumes that larger values of $m$ indicate better clustering quality. For some quality measures, smaller values indicate better clustering quality - in which case we reverse the direction of inequalities for consistency and use Argmin instead of Argmax for richness.

## References

[1] Jon Kleinberg. "An Impossibility Theorem for Clustering." Advances in Neural Information Processing Systems (NIPS) 15, 2002.

[2] Glen W. Milligan. "A Monte-Carlo study of 30 internal criterion measures for cluster-analysis." Psychometrica 46, 187-195, 1981.

[3] J. Puzicha, T. Hofmann, and J. Buhmann. "Theory of Proximity Based Clustering: Structure Detection by Optimization," Pattern Recognition, 33(2000).

